# A Neural Net Model for Adaptive Control of Saccadic Accuracy by Primate Cerebellum and Brainstem

Paul Dean[a], John E. W. Mayhew and Pat Langdon

Department of Psychology[a] and Artificial Intelligence
Vision Research Unit, University of Sheffield,
Sheffield S10 2TN, England.

## Abstract

Accurate saccades require interaction between brainstem circuitry and the cerebellum. A model of this interaction is described, based on Kawato's principle of feedback-error-learning. In the model a part of the brainstem (the superior colliculus) acts as a simple feedback controller with no knowledge of initial eye position, and provides an error signal for the cerebellum to correct for eye-muscle nonlinearities. This teaches the cerebellum, modelled as a CMAC, to adjust appropriately the gain on the brainstem burst-generator's internal feedback loop and so alter the size of burst sent to the motoneurons. With direction-only errors the system rapidly learns to make accurate horizontal eye movements from any starting position, and adapts realistically to subsequent simulated eye-muscle weakening or displacement of the saccadic target.

## 1 INTRODUCTION

The use of artificial neural nets (ANNs) to control robot movement offers advantages in situations where the relevant analytic solutions are unknown, or where unforeseeable changes, perhaps as a result of damage or wear, are likely to occur. It is also a mode of control with considerable similarities to those used in biological systems. It may thus prove possible to use ideas derived from studies of ANNs in robots to help understand how the brain produces movements. This paper describes an attempt to do this for saccadic eye movements.

The structure of the human retina, with its small foveal area of high acuity, requires extensive use of eye-movements to inspect regions of interest. To minimise the time during which the retinal image is blurred, these saccadic refixation movements are very rapid - too rapid for visual feedback to be used in acquiring the target (Carpenter 1988). The saccadic control system must therefore know in advance the size of control signal to be sent to the eye muscles. This is a function of both target displacement from the fovea and initial eye-position. The latter is important because the eye-muscles and orbital tissues are elastic, so that more force is required to move the eye away from the straight-ahead position than towards it (Collins 1975).

Similar rapid movements may be required of robot cameras. Here too the desired control signal is usually a function of both target displacement and initial camera positions. Experiments with a simulated four degree-of-freedom stereo camera rig have shown that appropriate ANN architectures can learn this kind of function reasonably efficiently (Dean et al. 1991), provided the nets are given accurate error information. However, this information is only available if the relevant equations have been solved; how can ANNs be used in situations where this is not the case?

A possible solution to this kind of problem (derived in part from analysis of biological motor control systems) has been suggested by Kawato (1990), and was implemented for the simulated stereo camera rig (Fig 1). Two controllers are arranged in

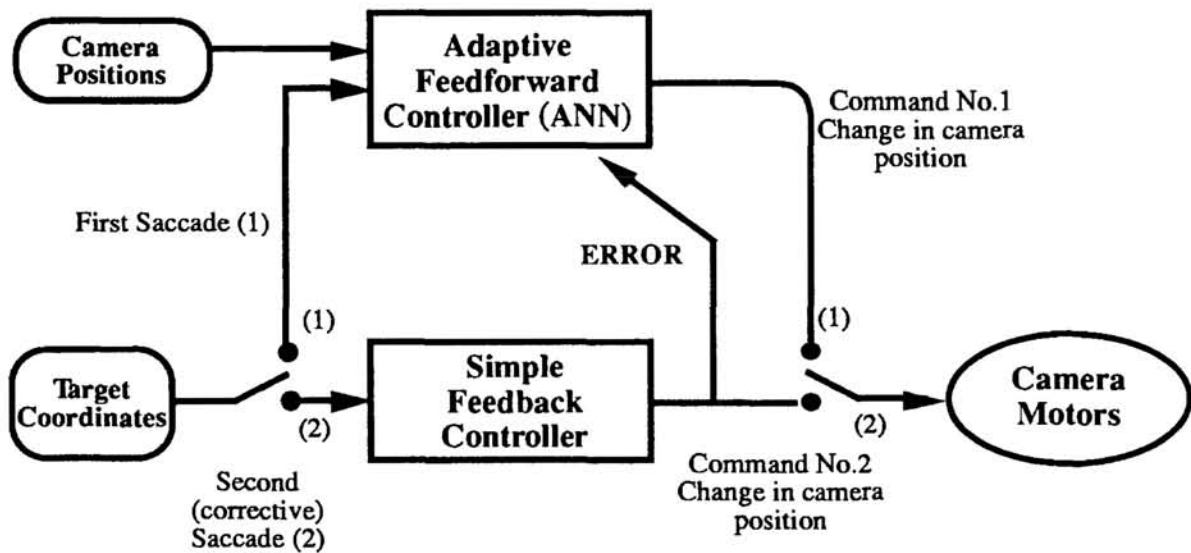

Fig 1:  Control architecture for robot saccades

parallel. Target coordinates, together with information about camera positions, are passed to an adaptive feedforward controller in the form of an ANN, which then moves the cameras. If the movement is inaccurate, the new target coordinates are passed to the second controller. This knows nothing of initial camera position, but issues a corrective movement command that is simply proportional to target displacement. In the absence of the adaptive controller it can be used to home in on the target with a series of saccades:

though each individual saccade is ballistic, the sequence is generated by visual feedback, hence the term simple feedback controller. When the adaptive controller is present, however, the output of the simple feedback controller can be used not only to generate a corrective saccade but also as a motor error signal (Fig 1). Although this error signal is not accurate, its imperfections become less important as the ANN learns and so takes on more responsibility for the movement (for proof of convergence see Kawato 1990). The architecture is robust in that it learns on-line, does not require mathematical knowledge, and still functions to some extent when the adaptive controller is untrained or damaged.

These qualities are also important for control of saccades in biological systems, and it is therefore of interest that there are similarities between the architecture shown in Fig 1 and the structure of the primate saccadic system (Fig 2). The cerebellum is widely (though

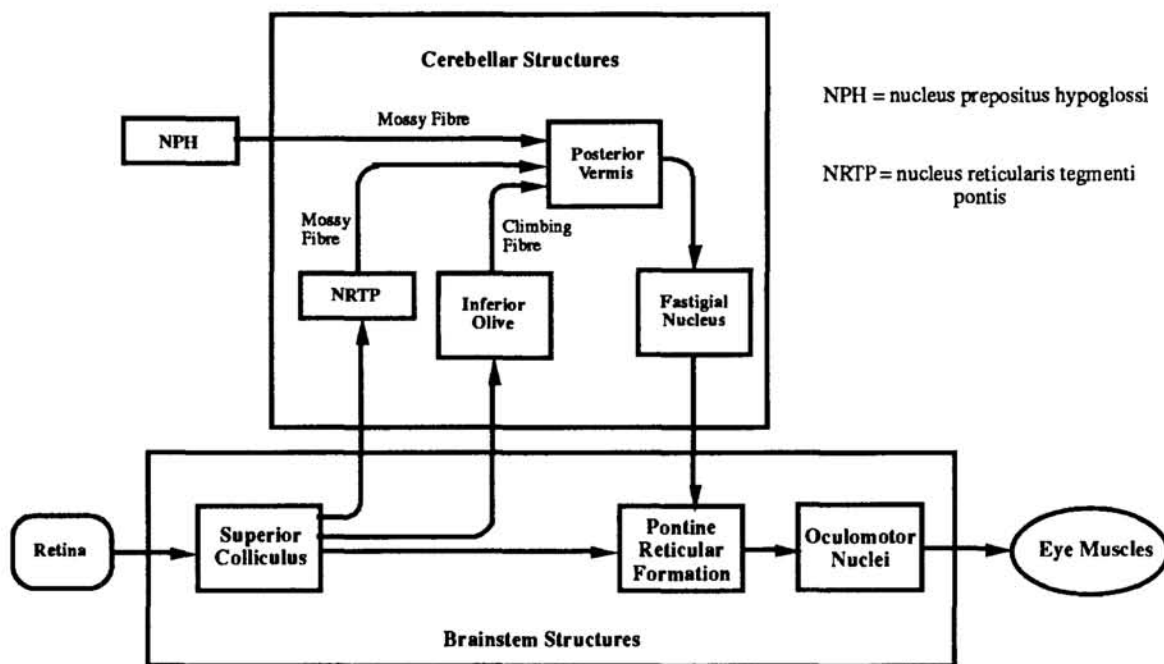

Fig 2: Schematic diagram of major components of primate saccadic control system

not universally) regarded as an adaptive controller, and when the relevant part of it is damaged the remaining brainstem structures function like the simple feedback controller of Fig 1. Saccades can still be made, but (i) they are not accurate; (ii) the degree of inaccuracy depends on initial eye position; (iii) multiple saccades are required to home in on the target; and (iv) the system never recovers (eg Ritchie 1976; Optican and Robinson 1980).

These similarities suggest that it is worth exploring the idea that the brainstem teaches the cerebellum to make accurate saccades (cf Grossberg and Kuperstein 1986), just as the simple feedback controller teaches the adaptive controller in the Kawato architecture. A model of the primate system was therefore constructed, using 'off-the-shelf' components wired together in accordance with known anatomy and physiology, and its performance assessed under a variety of conditions.

## 2  STRUCTURE OF MODEL

The overall structure of the model is shown in Fig 3. It has three main components: a simple feedback controller, a burst generator, and a CMAC. The simple feedback

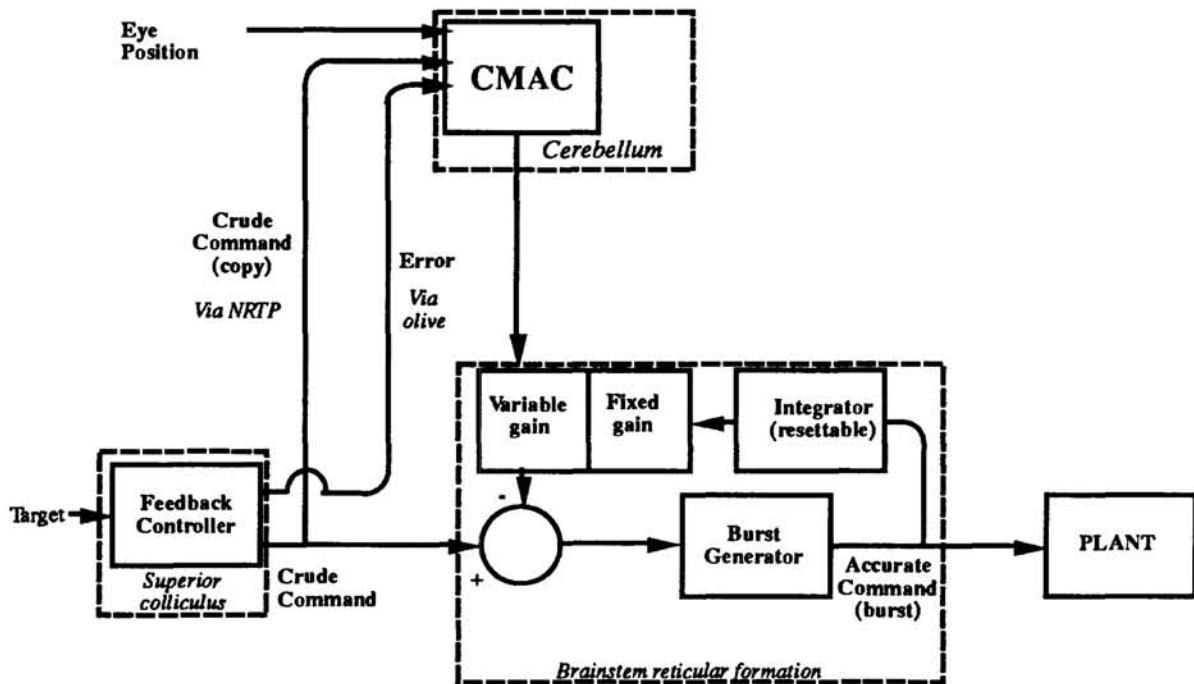

Figure 3: Main components of the model. The corresponding biological structures are shown in italics and dotted lines.

controller sends a signal proportional to target displacement from the fovea to the burst generator. The function of the burst generator is to translate this signal into an appropriate command for the eye muscles, and it is based here on the model of Robinson (Robinson 1975; van Gisbergen et al. 1981). Its output is a rapid burst of neural impulses, the frequency of which is esentially a velocity command. A crucial feature of Robinson's model is an internal feedback loop, in which the output of the generator is integrated and compared with the input command. The saccade terminates when the two are equal. This system ensures that the generator gives the output matching the input command in the face of disturbances that might alter burst frequency and hence saccade velocity.

The simple feedback controller sends to the CMAC (Albus 1981) a copy of its command to the burst generator. The CMAC (Cerebellar Model Arithmetic Computer) is a neural net model of the cerebellum incoporating theories of cerebellar function proposed independently by Marr (1969) and Albus (1971). Its function is to learn a mapping between a multidimensional input and a single-valued output, using a form of lookup table with local interpolation. The entries in the lookup table are modified using the delta rule, by an error signal which is either the difference between desired and actual output or some estimate of that difference. CMACs have been used successfully in a number of

applications concerning prediction or control (eg Miller et al. 1987; Hormel 1990). In the present case the function to be learnt is that relating desired saccade amplitude and initial eye position (inputs) to gain adjustment in the internal feedback loop of the burst generator (output).

The correspondences between the model structure and the anatomy and physiology of the primate saccadic system are as follows.
(1) The simple feedback controller represents the superior colliculus.
(2) The burst generator corresponds to groups of neurons located in the brainstem.
(3) The CMAC models a particular region of cerebellar cortex, the posterior vermis.
(4) The pathway conveying a copy of the feedback controller's crude command corresponds to the projection from the superior colliculus to the nucleus reticularis tegmenti pontis, which in turn sendes a mossy fibre projection to the posterior vermis.
Space precludes detailed evaluation of the substantial evidence supporting the above correspondences (see eg Wurtz and Goldberg 1989). The remaining two connections have a less secure basis.
(5) The idea that the cerebellum adjusts saccadic accuracy by altering feedback gains in the burst generator is based on stimulation evidence (Keller 1989); details of the projection, including its anatomy, are not known.
(6) The error pathway from feedback controller to CMAC is represented by the anatomically identified projection from superior colliculus to inferior olive, and thence via climbing fibres to the posterior vermis. There is considerable debate concerning the functional role of climbing fibres, and in the case of the tecto-olivary projection the relevant physiological evidence appears to be lacking.

## 3 PERFORMANCE OF MODEL

The system shown in Fig 3 was trained to make horizontal movements only. The size of burst $\Delta I$ (arbitrary units) required to produce an accurate rightward saccade $\Delta \theta$ deg was calculated from Van Gisbergen and Van Opstal's (1989) analysis of the nonlinear relationship between eye position and muscle position as

$$\Delta I = a \ [\Delta\theta^2 + \Delta\theta \ (b + 2\theta)] \qquad (1)$$

where $\theta$ is initial eye-position (measured in deg from extreme leftward eye-position) and a and b are constants. In the absence of the CMAC, the feedback controller and burst generator produce a burst of size

$$\Delta I \ = \ x. \ (c/d) \qquad (2)$$

where x is the rightward horizontal displacement of the target, c is the gain constant of the feedback controller, and d a constant related to the fixed gain of the internal feedback loop of the burst generator. The kinematics of the eye are such that x (measured in deg of visual angle) is equal to $\Delta\theta$. The constants were chosen so that the performance of the system without the CMAC resembled that of the primate saccadic system after cerebellar damage (fig 4A), namely position-dependent overshoot (eg Ritchie 1976; Optican and

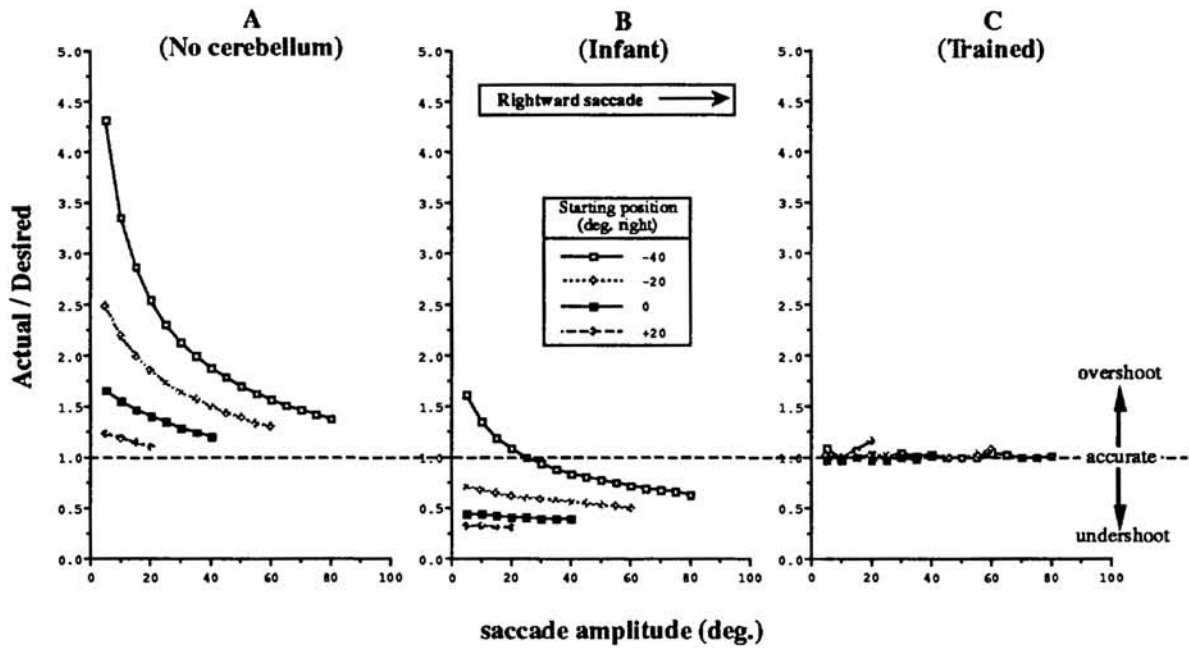

Fig 4. Performance of model under different conditions before and after training

Robinson 1980). When the CMAC is present, the size of burst changes to

$$\Delta I \quad = \quad x. \quad [c/(g + d)] \tag{3}$$

where g is the output of the CMAC. This was initialised to a value that produced a degree of saccadic undershoot (Fig 4b) characteristic of initial performance in human infants (eg Aslin 1987).

Training data were generated as 50,000 pairs of random numbers representing the initial position of the eye and the location of the target respectively. Each pair had to satisfy the constraints that (i) both lay within the oculomotor range (45 deg on either side of midline) and (ii) the target lay to the right of the starting position. For the test data the starting position varied from 40 deg left to 30 deg right in 10 degree steps. For each starting position there was a series of targets, starting at 5 deg to the right of the start and increasing in 5 degree steps up to 40 deg to the right of midline (a subset of the test data was used in Fig 4). The main measure of performance was the absolute gain error (ie the the difference between the actual gain and 1.0, always taken as positive) averaged over the test set.

The configuration of the CMAC was examined in pilot experiments. The CMAC coarse-codes its inputs, so that for a given resolution r, an input span of s can be represented as set of m measurement grids each dividing the input span into n compartments, where s/r = m.n. Combinations of m and n were examined, using perfect error feedback. A reasonable compromise between learning speed and asymptotic accuracy was achieved by using 10 coarse-coding grids each with 10x10 resolution (for the two input dimensions), giving a total of 1000 memory cells.

The main part of the study investigated first the effects of degrading the quality of the error feedback on learning. The main conclusion was that efficient learning could be obtained if the CMAC were told only the direction of the error, ie overshoot versus undershoot. This information was used to increase by a small fixed amount the weights in the activated cells (thereby producing increased gain in the internal feedback loop) when the saccade was too large, and to decreasing them similarly when it was too small. Appropriate choice of learning rate gave a realistic overall error of 5% (Fig 4c) after about 2000 trials. Direct comparison with learning rates of human infants, who take several months to achieve accuracy, is confounded by such factors as the maturation of the retina (Aslin 1987).

Learning parameters were then kept constant, and the model tested with simulations of two different conditions that produce saccadic plasticity in adult humans. One involved the effects of weakening the rightward pulling eye muscle in one eye. In people, the weakened eye can be trained by covering the normal eye with a patch, an effect which experiments with monkeys indicate depends on the cerebellum (Optican and Robinson 1980). For the model eye-weakening was simulated by increasing the constant a in equation (1) such that the trained system gave an average gain of about 0.5. Retraining required about 400-500 trials. Testing the previously normal eye (ie with the original value of a) showed that it now overshot, as is also the case in patients and experimental animals. Again normal performance was restored after 400-500 trials. These learning rates compare favourably with those observed in experimental animals.

Finally, the second simulation of adult saccadic plasticity concerned the effects of moving the target during a saccade. If the target is moved in the opposite direction to its original displacement the saccade will overshoot, but after a few trials adaptation occurs and the saccade becomes 'accurate' once more. Simulation of the procedure used by Deubel et al. (1986) gave system adaptation rates similar to those observed experimentally in people.

## 4   CONCLUSIONS

These results indicate that the model can account in general terms for the acquisition and maintenance of saccadic accuracy in primates (at least in one dimension). In addition to its general biologically attractive properties, the model's structure is consistent with current anatomical and physiological knowledge, and offers testable predictions about the functions of the hitherto mysterious projections from superior colliculus to posterior vermis. If these predictions are supported by experimental evidence, it would be appropriate to extend the model to incorporate greater physiological detail, for example concerning the precise location(s) of cerebellar plasticity.

**Acknowledgements**

This work was supported by the Joint Council Initiative in Cognitive Science.

## References

Albus, J.A. (1971) A theory of cerebellar function. *Math. Biosci.* 10: 25-61.

Albus, J.A. (1981) *Brains, Behavior and Robotics.* BYTE books (McGraw-Hill), Peterborough New Hampshire.

Aslin, R.N. (1987) Motor aspects of visual development in infancy. In: *Handbook of Infant Perception*, eds. P. Salapatek and L. Cohen. Academic Press, New York, pp.43-113.

Collins, C.C. (1975) The human oculomotor control system. In: *Basic Mechanisms of Ocular Motility and their Clinical Implications*, eds. G. Lennerstrand and P. Bach-y-Rita. Pergamon Press, Oxford, pp. 145-180.

Dean, P., Mayhew, J.E.W., Thacker, T. and Langdon, P. (1991) Saccade control in a simulated robot camera-head system: neural net architectures for efficient learning of inverse kinematics. *Biol. Cybern.* 66: 27-36.

Deubel, H., Wolf, W. and Hauske, G. (1986) Adaptive gain control of saccadic eye movements. *Human Neurobiol.* 5: 245-253.

Grossberg, S. and Kuperstein, M. (1986) *Neural Dynamics of Adaptive Sensory-Motor Control: Ballistic Eye Movements.* Elsevier, Amsterdam.

Hormel, M. (1990) A self-organising associative memory system for control applications. In: *Advances in Neural Information Processing Systems 2*, ed. D.S. Touretzky. Morgan Kaufman, San Mateo, California, pp. 332-339.

Kawato, M. (1990) Feedback-error-learning neural network for supervised motor learning. In *Advanced Neural Computers*, ed. R. Eckmiller. Elsevier, Amsterdam, pp.365-372.

Keller, E.L. (1989) The cerebellum. In: *The Neurobiology of Saccadic Eye Movements*, eds. Wurtz, R.H. and Goldberg, M.E. Elsevier Science Publishers, North Holland, pp. 391-411.

Marr, D. (1969) A theory of cerebellar cortex. *J. Physiol.* 202: 437-470.

Miller, W.T. III, Glanz, F.H. and Gordon Kraft, L. III (1987) Application of a general learning algorithm to the control of robotic manipulators. *Int. J. Robotics Res.* 6: 84-98.

Optican, L.M. and Robinson, D.A. (1980) Cerebellar-dependent adaptive control of primate saccadic system. *J. Neurophysiol.* 44: 1058-1076.

Ritchie, L. (1976) Effects of cerebellar lesions on saccadic eye movements. *J. Neurophysiol.* 39: 1246-1256.

Robinson, D.A. (1975) Oculomotor control signals. In: *Basic Mechanisms of Ocular Motility and their Clinical Implications*, eds. Lennerstrand, G. and Bach-y-Rita, P. Pergamon Press, Oxford, pp. 337-374.

Van Gisbergen, J.A.M., Robinson, D.A. and Gielen, S. (1981) A quantitative analysis of generation of saccadic eye movements by burst neurons. *J. Neurophysiol.* 45: 417-442.

Van Gisbergen, J.A.M. and van Opstal, A.J. (1989) Models. In: *The Neurobiology of Saccadic Eye Movements*, eds. Wurtz, R.H. and Goldberg, M.E. Elsevier Science Publishers, North Holland, pp. 69-101.

Wurtz, R.H. and Goldberg, M.E. (1989) *The Neurobiology of Saccadic Eye Movements.* Elsevier Science Publishers, North Holland.
